# Mean-Field Approach to a Probabilistic Model in Information Retrieval

**Bin Wu, K. Y. Michael Wong**
Department of Physics
Hong Kong University of Science and Technology
Clear Water Bay, Hong Kong
phwbd@ust.hk    phkywong@ust.hk

**David Bodoff**
Department of ISMT
Hong Kong University of Science and Technology
Clear Water Bay, Hong Kong
dbodoff@ust.hk

## Abstract

We study an explicit parametric model of documents, queries, and relevancy assessment for Information Retrieval (IR). Mean-field methods are applied to analyze the model and derive efficient practical algorithms to estimate the parameters in the problem. The hyperparameters are estimated by a fast approximate leave-one-out cross-validation procedure based on the cavity method. The algorithm is further evaluated on several benchmark databases by comparing with standard algorithms in IR.

## 1   Introduction

The area of information retrieval (IR) studies the representation, organization and access of information in an information repository. With the advent and boom of the Internet, especially the World Wide Web (WWW), more and more information is available to be shared online. Search on the Internet becomes increasingly popular. In this respect, probabilistic models have become very useful in empowering information searches [1, 2].

In fact, information searches themselves contain rich information, which can be recorded and fruitfully used to improve the performance of subsequent retrievals. This is an extension of the process of *relevance feedback* [3], which incorporates the relevance assessments supplied by the user to construct new representations for queries, during the procedure of the users interactive document retrieval. In the process, the feedback information helps to refine the queries continuously, but the effects pertain only to the particular retrieval session. On the other hand, our objective is to refine the representations of documents and queries with the help of relevancy data, so that subsequent retrieval sessions can be benefited.

Based on Fuhr and Buckley's meta-structure [4] relating documents, queries and relevancy assessments, one of us recently proposed a probabilistic model [5] in which these objects

are described by explicit parametric distribution functions, facilitating the construction of a likelihood function, whose maximum can be used to characterize the documents and queries. Rather than relying on heuristics as in many previous work, the proposed model provides a unified formal framework for the following two tasks: (a) *ad hoc* information retrieval, in which a query is given and the goal is to return a list of ranked documents according to their similarities with the query; (b) document routing, in which a document is given and the goal is to categorize it using a list of ranked queries according to their similarities with the document. (Here we assume a model in which categories are represented by queries.)

In this paper, we report our recent progress in putting this new theoretical approach to empirical tests. Since documents and queries are represented by high dimensional vectors in a vector space model, a mean-field approach will be adopted. mean-field methods were commonly used to study magnetic systems in statistical physics, but thanks to their ability to deal with high dimensional systems, they are increasingly applied to many areas of information processing recently [6]. In the present context, a mean-field treatment implies that when a particular component of a document or query vector is analyzed, all other components of the same and other vectors can be considered as background fields satisfying appropriate average properties, and correlations of statistical fluctuations with the background vectors can be neglected.

After introducing the parametric model in Section 2, the mean-field approach will be used in two steps. First, in Section 3, the *true* representations of documents and queries will be estimated by maximizing the total probability of observation. It results in a set of mean-field equations, which can be solved by a fast iterative algorithm. Respectively, the estimated true documents and queries will then be used for *ad hoc* information retrieval and document routing.

Secondly, the model depends on a few hyperparameters which are conventionally determined by the cross-validation method. Here, as described in Section 4, the mean-field approach can be used again to accelerate the otherwise tedious leave-one-out cross-validation procedure. For a given set of hyperparameter values, it enables us to carry out the systemwide iteration only once (rather than repeating once for each left-out document or query), and the leave-one-out estimations of the document and query representations can be obtained by a version of mean-field theory called the cavity method [7].

In Section 6, we compare the model with the standard *tf-idf* [8] and latent semantic indexing (LSI) [9] on benchmark test collections. As we shall see, the validity of our model is well supported by its superior performance. The paper is concluded in Section 7.

## 2   A Unified Probabilistic Model

Our work is motivated by Fuhr and Buckley's conceptual model. Assume that a set of $N_d$ documents and $N_q$ queries is available to us. In the vector space model, each document and query is represented by an $M$ dimensional vector. The vectors are denoted by $D$ ( $Q$ ), which are referred to as the *true meaning* of the document (query). Our model consists of the following 3 components:

(a) The document $D^0$ we really observe is distributed around the true document vector $D$ according to the probability distribution $f_D(D^0|D)$, the difference resulting from the documents containing terms that do not ideally represent the meaning of the document. In other words, the document $D^0$ is generated from its true meaning $D$.

(b) Similarly, the query $Q^0$ that the user actually submits is also distributed around the true query vector $Q$ according to the probability distribution distribution $f_Q(Q^0|Q)$.

(c) There is some relation between the document and query, called relevancy assessment. We denote this relation with a binary variable $B$ for each pair of document and query. If $B = 1$, we say the document is relevant to the query, that is, the document is what the user wants. Otherwise, $B = 0$ and the document is irrelevant to the query. Suppose we have some relevancy relations between documents and queries (through historical records, from experts, etc.). Then we hypothesize that the true documents and queries are distributed according to the distribution $f_B(D, Q|B)$, that is, the true representation of documents and queries should satisfy their relevancy relations.

We summarize the idea through a probabilistic meta-structure shown in Figure 1.

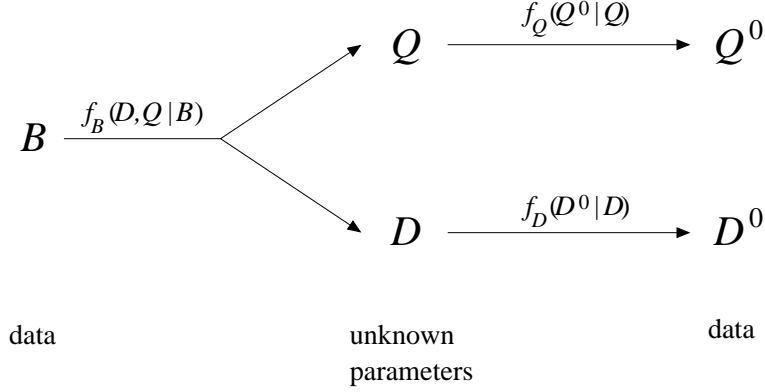

Figure 1: Probabilistic meta-structure

In order to complete the model, we need to hypothesize the form of the distribution functions. In this paper, we restrict the documents and queries to a hypersphere, since usually only the cosines of the angles between documents and queries are used to determine the similarity between documents and queries. Hence, we assume the following distribution functions:

(a) The distribution of each observed document $D^0$ given its true location $D$:

$$f_D(D^0|D; \alpha_d) = \frac{\exp(\alpha_d D \cdot D^0)\delta(\|D^0\|^2 - 1)}{Z_D}. \tag{1}$$

(b) The distribution of each observed query $Q^0$ given its true location $Q$:

$$f_Q(Q^0|Q; \alpha_q) = \frac{\exp(\alpha_q Q \cdot Q^0)\delta(\|Q^0\|^2 - 1)}{Z_Q}. \tag{2}$$

(c) The prior distribution of the documents and queries, given the relevance relation between them:

$$f_B(\{D_i, Q_j\}|B; \beta) = \frac{\prod_{i,j} \exp(\beta B_{ij} D_i \cdot Q_j)\delta(\|(D_i\|^2 - 1)\delta(\|Q_j\|^2 - 1))}{Z^B}, \tag{3}$$

where $\delta(x)$ is the Dirac $\delta$-function, and $Z_D$, $Z_Q$ and $Z_B$ are normalization constants of $f_D$, $f_Q$ and $f_B$ respectively, and are hence independent of $D$ and $Q$.

If we further assume that the observation of documents and queries are independent of each other, we can obtain the total probability of observing all documents and queries, given the relevancy relation between them:

$$P(\{D_i^0, Q_j^0\}|B; \Xi) = \frac{Z_L}{Z_B(Z_D)^{N_D}(Z_Q)^{N_Q}}\delta(\|D_i^0\|^2 - 1)\delta(\|Q_j^0\|^2 - 1), \tag{4}$$

where

$$Z_L = \int \prod_{i,j} \mathrm{d}D_i \mathrm{d}Q_j \delta(\|D_i\|^2 - 1)\delta(\|Q_j\|^2 - 1)\exp(-\beta E),$$ (5)

$$\beta E = -\beta \sum_{i,j} B_{ij} D_i \cdot Q_j - \alpha_d \sum_i D_i^0 \cdot D_i - \alpha_q \sum_j Q_j^0 \cdot Q_j,$$ (6)

and $\Xi$ denotes all hyperparameters $\{\alpha_d, \alpha_q, \beta\}$. There is now an appealing correspondence between the present model and spin models in statistical physics. It is observed that $Z_L$ is just the familiar partition function and $E$ is the energy function.

By maximizing the probability in Eq. (4), we can obtain an estimation of the true documents $\hat{D}$, which can be used in *ad hoc retrieval*: we define the similarity function between two vectors as the cosine of the angle between them, and rank the similarities between $\hat{D}$ (instead of $D^0$) with a new query to determine whether the documents should be retrieved or not. As a byproduct, we can also obtain the estimation of the true queries $\hat{Q}$ , which in turn can be used in document routing: new documents should be compared with $\hat{Q}$ to determine whether it belongs to this category or not. So our model gives a unifying procedure for both *ad hoc* retrieval and routing.

## 3   Parameter Estimation

In this section, we derive a fast iterative algorithm for parameter estimation. First, we replace the $\delta$-function by its Fourier transform. Then $Z_L$ can be written as

$$Z_L = \int_{-\mathrm{i}\infty}^{+\mathrm{i}\infty} \prod_i \frac{\mathrm{d}\mu_i}{2\pi\mathrm{i}} \prod_j \frac{\mathrm{d}\nu_j}{2\pi\mathrm{i}} \int \prod \mathrm{d}D_i \mathrm{d}Q_j \exp(-F),$$ (7)

where $F(D_i, Q_j, \mu_i, \nu_j) = \sum_i \mu_i(\|D_i\|^2 - 1) + \sum_j \nu_j(\|Q_j\|^2 - 1) + \beta E$. In writing this formula, we have changed the integration to the imaginary axis.

Mean-field theory works in the limit of large $N_d$, $N_q$ and $M$, when the integration can be well approximated by taking the saddle point of $F$. This is obtained by equating the partial derivatives of $F$ with respect to $D$, $Q$, $\mu$ and $\nu$ to zero, yielding

$$\hat{D}_i = \frac{\beta \sum_j B_{ij} \hat{Q}_j + \alpha_d D_i^0}{2\hat{\mu}_i},$$ (8)

$$\hat{Q}_j = \frac{\beta \sum_i B_{ij} \hat{D}_i + \alpha_q Q_j^0}{2\hat{\nu}_j},$$ (9)

$$\hat{\mu}_i = \frac{1}{2}\|\beta \sum_j B_{ij} \hat{Q}_j + \alpha_d D_i^0\|,$$ (10)

$$\hat{\nu}_j = \frac{1}{2}\|\beta \sum_i B_{ij} \hat{D}_i + \alpha_q Q_j^0\|.$$ (11)

This set of equations is referred to as the mean-field equations, since fluctuations around the mean values of the parameters have been neglected. Due to its simple form, it can be solved by an iterative scheme. Though we have not studied the theoretical convergence of the iterative scheme, its effectiveness can be seen from the following arguments. If we replace $\hat{\mu}_i$ in Eq. (8) and $\hat{\nu}_j$ in Eq. (9) by the respective values of $\mu_i^*$ and $\nu_j^*$ at the saddle point, then the iteration process becomes a linear one. Now, Eqs. (8) and (9) differ from this linear iteration problem by scale factors of $\mu_i^*/\hat{\mu}_i$ and $\nu_j^*/\hat{\nu}_j$ respectively. Hence after using Eqs. (10) and (11), the problem is equivalent to rescaling the lengths of the iterated

vectors back to the hypersphere defined by $\|D_i\|^2 = 1$ and $\|Q_j\|^2 = 1$. This alternate operation of linear iteration and rescaling back to the hypersphere makes it a very stable algorithm. The complexity of the algorithm is linear in the number of documents and queries. Empirically, it converges in just a few tens of steps. Alternatively, one may use the Augmented Lagrangian method to find the saddle point of $F$, whose convergence is guaranteed, but is computationally more complex [10].

## 4   Hyperparameter Estimation

In our model, the parameters $\beta$, $\alpha_d$ and $\alpha_q$ determine the shape of the distributions $f_D$, $f_Q$ and $f_B$, and influence the parameter estimation described in Section 3. We refer to them as hyperparameters. They have to be chosen so that the model performs optimally when new queries are raised to retrieve documents, or when new documents are routed.

A standard method for hyperparameter estimation in machine learning is leave-one-out cross-validation [11]. Suppose we have $N$ examples for training the model. Then each time we pick one data as the validation set and train the model with the rest of the $N-1$ examples. The hyperparameters are chosen as the ones that give the optimal performance averaged over the $N$ test examples.

The exact leave-one-out cross-validation is very tedious, especially for multiple hyperparameters, because of the need to train the model $N$ times for each combination of hyperparameters. For this model, we propose an approximate leave-one-out procedure based on the cavity method [7]. Suppose we have trained the model with all data, and obtain the estimation $\{\hat{D}_i, \hat{Q}_j\}$, which satisfies the steady state equation

$$\hat{D}_i = \frac{\beta \sum_j B_{ij}\hat{Q}_j + \alpha_d D_i^0}{2\hat{\mu}_i}, \qquad \hat{Q}_j = \frac{\beta \sum_i B_{ij}\hat{D}_i + \alpha_q Q_j^0}{2\hat{\nu}_j}. \tag{12}$$

If the query $Q_b$ were left out from the training set of queries, the cavity estimation should satisfy the equation

$$\hat{D}_i^{\setminus b} = \frac{\beta \sum_{j\neq b} B_{ij}\hat{Q}_j^{\setminus b} + \alpha_d D_i^0}{2\hat{\mu}_i^{\setminus b}}, \qquad \hat{Q}_j^{\setminus b} = \frac{\beta \sum_i B_{ij}\hat{D}_i^{\setminus b} + \alpha_q Q_j^0}{2\hat{\nu}_j^{\setminus b}}, \quad j \neq b. \tag{13}$$

By subtracting (7) by (8), and assuming that $\{\mu_i^{\setminus b}, \nu_j^{\setminus b}\}$ is approximately the same as $\{\mu_i, \nu_j\}$, we can get the difference,

$$\Delta D_i \approx \frac{\beta \sum_{j\neq b} B_{ij}\Delta Q_j + \beta B_{ib}Q_b^0}{2\hat{\mu}_i}, \qquad \Delta Q_j \approx \frac{\beta \sum_i B_{ij}\Delta D_i}{2\hat{\nu}_j}, \quad j \neq b. \tag{14}$$

For *ad hoc* retrieval, we eliminate $\Delta Q_j$ to obtain a set of linear equations for $\Delta D_i$. The solution can be further simplified by using the mean-field argument that the changes induced by removing the query $Q_b$ on documents $i$ can be decoupled. Hence we can neglect the off-diagonal terms, yielding

$$\Delta D_i = \frac{\beta B_{ib}Q_b^0}{2\hat{\mu}_i - \beta^2 \sum_{j\neq b} \frac{B_{ij}^2}{2\hat{\nu}_j}}. \tag{15}$$

Note that $\{\hat{\mu}_i, \hat{\nu}_j\}$ have been known in the systemwide training. Then $\hat{D}_i^{\setminus b}$ can be estimated by $\hat{D}_i^{\setminus b} = D_i - \Delta D_i$. The similarities between $Q_b$ and $\hat{D}_i^{\setminus b}$ are then used to predict the leave-one-out *ad hoc* retrieval performance of the model. Equations for document routing can be derived analogously.

Note that we need to train the model only once, and the leave-one-out estimation of documents and queries can be obtained in one step. So the algorithm is extremely fast. Amazingly, it also gives reasonable estimations of hyperparameters, as shown in the following experiments.

We remark that the mean-field technique can be applied to distributions of documents, queries and relevance feedbacks other than those described by Eqs. (1-3). In the present case spectified by Eqs. (1-3), our model is similar to the Gaussian model, if the spherical constraint on $Q$'s and $D$'s are replaced by a spherical Gaussian prior. Though leave-one-out cross-validation can be done exactly in the Gaussian model, it involves the inversion of a large matrix. On the other hand, the mean-field estimation greatly simplifies the process by neglecting the off-diagonal elements.

## 5   Experimental Results

We have applied the proposed method to *ad hoc* retrieval and routing for the test collections of Cranfield and CISI. Because we treat both tasks identically, we use the same evaluation criterion: the recall precision curve and the average retrieval precision. We have run two versions of our algorithm: (a) in the original dimension, the observed documents $D^0$ and queries $Q^0$ are represented by the original *tf-idf* weights; (b) in the reduced dimension of 100, in which the original vectors are reduced by singular value decomposition (SVD) in LSI.

In Figs. 2 (a-b), we show the recall precision curves at the optimal hyperparameters. The mean-field estimates are compared with the baseline results of LSI. It is clear that our method gives significant gains in retrieval precision. Comparisons using the original dimension or the Cranfield collection, not shown here due to space limitations, yield equally satisfactory results.

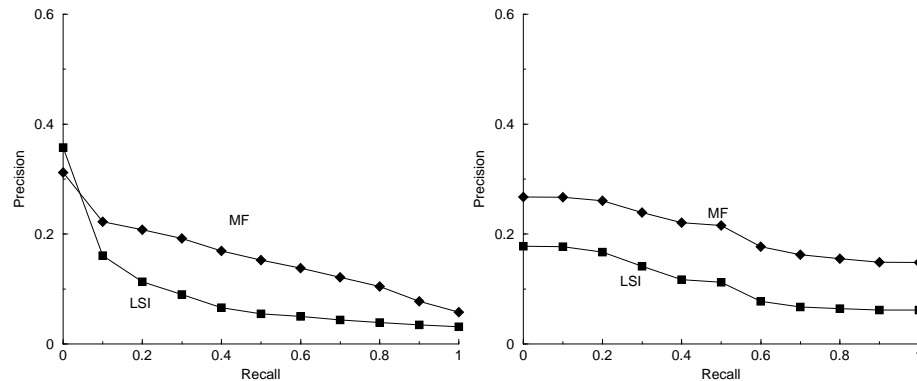

Figure 2: The recall precision curves of the mean-field estimation (MF) and the baseline (LSI) for (a) *ad hoc* retrieval (b) document routing for CISI in reduced dimension

For hyperparameter estimation, we can compare the mean-field results and those for exact leave-one-out cross-validation in reduced dimension, since the computation of the exact ones is still feasible. In Fig. 3, we have plotted the average precision versus the two hyperparameters, as computed by the two methods. They have very similar contours, although there is a uniform displacement between their values. This demonstrates the usefulness of the mean-field approximation in hyperparameter estimation.

In Table 1, we obtain the values of the optimal hyperparameters from the mean-field leave-

one-out method, and the average precisions of the exact leave-one-out are then computed using these optimal hyperparameters. These are compared with the results of the exact leave-one-out and listed in Table 1. For the hyperparameter estimation in the original dimension, the exact leave-one-out is not available since it is too tedious. Instead, we compare the hyperparameters with the ones from the $k$-fold cross-validation. Whether we compare the mean-field with the exact leave-one-out or $k$-fold cross-validation, the optimal hperparameters are comparable in most cases, and when there are discrepancies, one can observe that the average precisions are essentially the same.

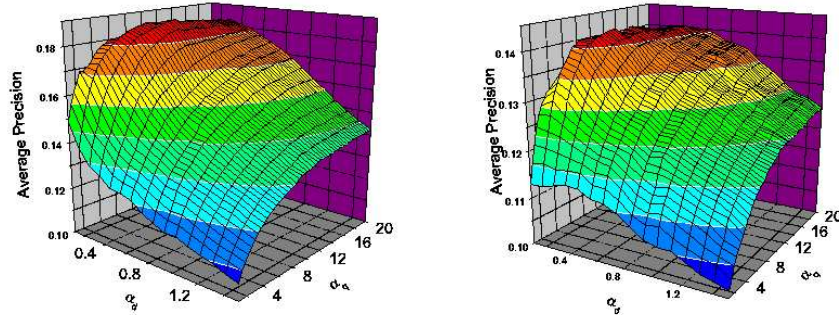

Figure 3: Average retrieval precision versus hyperparameters for *ad hoc* retrieval in reduced dimension for CISI: (a) mean-field leave-one-out, peaked at $(\alpha_d/\beta, \alpha_q/\beta) = (0.3, 12.0)$; (b) exact leave-one-out. peaked at $(\alpha_d/\beta, \alpha_q/\beta) = (0.3, 10.1)$.

Table 1: The average retrieval precision for leave-one-out cross-validation in reduced dimension: mean-field versus exact.

|  | CISI | | | Cranfield | | |
|---|---|---|---|---|---|---|
|  | $\alpha_d/\beta$ | $\alpha_q/\beta$ | Average precision | $\alpha_d/\beta$ | $\alpha_q/\beta$ | Average precision |
|  | *ad hoc* retrieval | | | | | |
| LSI | – | – | 0.079 | – | – | 0.178 |
| Mean-Field | 0.3 | 12.0 | 0.142 | 0.4 | 1.1 | 0.248 |
| Exact | 0.3 | 10.1 | 0.142 | 0.6 | 1.5 | 0.250 |
|  | Document Routing | | | | | |
| LSI | – | – | 0.104 | – | – | 0.240 |
| Mean-Field | 28.9 | 1.6 | 0.192 | 2.5 | 1.1 | 0.351 |
| Exact | 23.0 | 2.5 | 0.193 | 0.9 | 0.7 | 0.356 |

## 6 Conclusion

We have considered a probabilistic model of documents, queries and relevancy assessments. Fast algorithms are derived for parameter and hyperparameter estimations. Significant improvement is achieved for both *ad hoc* retrieval and routing compared with *tf-idf* and LSI. In another paper [12], we have compared the model with other heuristic methods such as Rocchio heuristics [3] and Bartell's Multidimensional Scaling [13], and the mean-field method still outperforms them. These successes illustrate the potentials of the mean-field approach, which is especially suitable for systems with high dimensions and

numerous mutually interacting components, such as those in IR. Hence we anticipate that mean-field methods will have increasing applications in many other probabilistic models in IR.

**Acknowledgments**

We thank R. Jin for interesting discussions. This work was supported by the grant HKUST6157/99P of the Research Grant Council of Hong Kong.

# References

[1] Cohn, D. and T. Hofmann (2001). The Missing Link – A Probabilistic Model of Document Content and Hypertext Connectivity. *Advances in Neural Information Processing Systems* **13**, T. K. Leen, T. G. Dietterich and V. Tresp, eds., MIT Press, Cambridge, MA, 430-436.

[2] Jaakola, T. and H. Siegelmann (2002). Active Information Retrieval. *Advances in Neural Information Processing Systems* **14**, T. G. Dietterich, S. Becker and Z. Ghahramani, eds., MIT Press, Cambridge, MA, 777-784.

[3] Rocchio, J. J. (1971). Relevance Feedback in Information Retrieval. *SMART Retrieval System–Experiments in Automatic Document Processing*, G. Salton ed., Prentice-Hall, Englewood Cliffs, NJ, Chapter 14.

[4] Fuhr, N. and C. Buckley (1991). A Probabilistic Learning Approach for Document Indexing. *ACM Transactions on Information Systems* **9**(3): 223-248.

[5] Bodoff, D., D. Enabe, A. Kanbil, G. Simon and A. Yukhimets (2001). A Unified Maximumn Likelihood Approach to Document Retrieval. *Journal of the American Society for Information Science and Technology* **52**(10): 785-796.

[6] Opper, M. and D. Saad, eds. (2001). *Advanced Mean Field Methods*, MIT Press, Cambridge, MA.

[7] Wong, K. Y. M. and F. Li (2002). Fast Parameter Estimation Using Green's Functions. *Advances in Neural Information Processing System* **14**: 535-542, T.G. Dietterich, S. Becker and Z. Ghahramani, eds., MIT Press, Cambridge, MA.

[8] Salton, G. and M. J. McGill (1983). *Introduction to Modern Information Retrieval*, McGraw-Hill, New York, 63-66.

[9] Deerwester, S., S. T. Dumais, G. W. Furnas, T. K. Landauer and R. Harshman (1990). Indexing by Latent Semantic Analysis. *Journal of the American Society for Information Science* **41**(16): 391-407.

[10] Nocedal, J. and S. J. Wright (1999). *Numerical Optimization*, Springer, Berlin, Ch. 17.

[11] Bishop, C. M. (1995). *Neural Networks for Pattern Recognition*, Clarendon Press, Oxford, 372-375.

[12] Bodoff, D., B. Wu and K. Y. M. Wong (2002). Relevance Feedback meets Maximum Likelihood, preprint.

[13] Bartell, B. T., G. W. Cottrell and R. K. Belew (1992). Latent Semantic Indexing Is an Optimal Special Case of Multidimensional Scaling. *Proceedings of the 15th International ACM SIGIR Conference on Research and Development in Information Retrieval*, 161-167.
